# REMAP: Recursive Estimation and Maximization of A Posteriori Probabilities - Application to Transition-Based Connectionist Speech Recognition

Yochai Konig, Hervé Bourlard,* and Nelson Morgan
{konig,bourlard,morgan}@icsi.berkeley.edu
International Computer Science Institute
1947 Center Street Berkeley, CA 94704, USA.

## Abstract

In this paper, we introduce REMAP, an approach for the training and estimation of posterior probabilities using a recursive algorithm that is reminiscent of the EM-based Forward-Backward (Liporace 1982) algorithm for the estimation of sequence likelihoods. Although very general, the method is developed in the context of a statistical model for transition-based speech recognition using Artificial Neural Networks (ANN) to generate probabilities for Hidden Markov Models (HMMs). In the new approach, we use local conditional posterior probabilities of transitions to estimate global posterior probabilities of word sequences. Although we still use ANNs to estimate posterior probabilities, the network is trained with targets that are themselves estimates of local posterior probabilities. An initial experimental result shows a significant decrease in error-rate in comparison to a baseline system.

## 1  INTRODUCTION

The ultimate goal in speech recognition is to determine the sequence of words that has been uttered. Classical pattern recognition theory shows that the best possible system (in the sense of minimum probability of error) is the one that chooses the word sequence with the maximum a posteriori probability (conditioned on the

evidence). If word sequence $i$ is represented by the statistical model $M_i$, and the evidence (which, for the application reported here, is acoustical) is represented by a sequence $X = \{x_1, \ldots, x_n, \ldots, x_N\}$, then we wish to choose the sequence that corresponds to the largest $P(M_i|X)$. In (Bourlard & Morgan 1994), summarizing earlier work (such as (Bourlard & Wellekens 1989)), we showed that it was possible to compute the global a posteriori probability $P(M|X)$ of a discriminant form of Hidden Markov Model (Discriminant HMM), $M$, given a sequence of acoustic vectors $X$. In Discriminant HMMs, the global a posteriori probability $P(M|X)$ is computed as follows: if $\Gamma$ represents all legal paths (state sequences $q_1, q_2, \ldots, q_N$) in $M_i$, $N$ being the length of the sequence, then

$$P(M_i|X) = \sum_\Gamma P(M_i, q_1, q_2, \ldots, q_N|X)$$

in which $q_n$ represents the specific state hypothesized at time $n$, from the set $Q = \{q^1, \ldots, q^\ell, q^k, \ldots, q^K\}$ of all possible HMM states making up all possible models $M_i$. We can further decompose this into:

$$P(M_i, q_1, q_2, \ldots, q_N|X) = P(q_1, q_2, \ldots, q_N|X)P(M_i|q_1, q_2, \ldots, q_N, X)$$

Under the assumptions stated in (Bourlard & Morgan 1994) we can compute

$$P(q_1, q_2, \ldots, q_N|X) = \prod_{n=1}^N p(q_n|q_{n-1}, x_n)$$

The Discriminant HMM is thus described in terms of *conditional transition probabilities* $p(q_n^\ell|q_{n-1}^k, x_n)$, in which $q_n^\ell$ stands for the specific state $q^\ell$ of $Q$ hypothesized at time $n$ and can be schematically represented as in Figure 1.

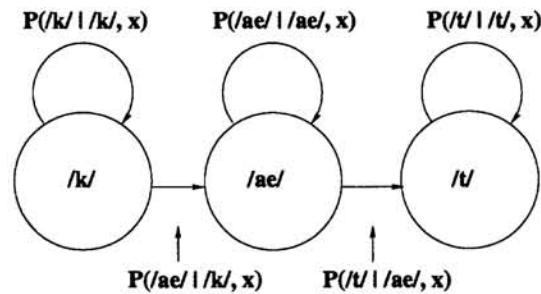

Figure 1: An example Discriminant HMM for the word "cat". The variable $x$ refers to a specific acoustic observation $x_n$ at time $n$.

Finally, given a state sequence we assume the following approximation:

$$P(M_i|q_1, q_2, \ldots, q_N, X) \approx P(M_i|q_1, q_2, \ldots, q_N)$$

We can estimate the right side of this last equation from a phonological model (in the case that a given state sequence can belong to two different models). All the required (local) conditional transition probabilities $p(q_n^\ell|q_{n-1}^k, x_n)$ can be estimated by the Multi-Layer Perceptron (MLP) shown in Figure 2.

Recent work at ICSI has provided us with further insight into the discriminant HMM, particularly in light of recent work on transition-based models (Konig & Morgan 1994; Morgan *et al.* 1994). This new perspective has motivated us to further develop the original Discriminant HMM theory. The new approach uses posterior probabilities at both local and global levels and is more discriminant in nature. In this paper, we introduce the Recursive Estimation-Maximization of A posteriori

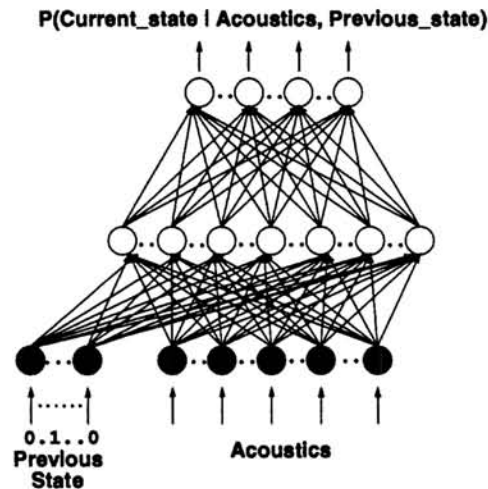

Figure 2: An MLP that estimates local conditional transition probabilities.

Probabilities (REMAP) training algorithm for hybrid HMM/MLP systems. The proposed algorithm models a probability distribution over all possible transitions (from all possible states and for all possible time frames $n$) rather than picking a single time point as a transition target. Furthermore, the algorithm incrementally increases the posterior probability of the correct model, while reducing the posterior probabilities of all other models. Thus, it brings the overall system closer to the optimal Bayes classifier.

A wide range of discriminant approaches to speech recognition have been studied by researchers (Katagiri *et al.* 1991; Bengio *et al.* 1992; Bourlard *et al.* 1994). A significant difficulty that has remained in applying these approaches to continuous speech recognition has been the requirement to run computationally intensive algorithms on all of the rival sentences. Since this is not generally feasible, compromises must always be made in practice. For instance, estimates for all rival sentences can be derived from a list of the "N-best" utterance hypotheses, or by using a fully connected word model composed of all phonemes.

## 2   REMAP TRAINING OF THE DISCRIMINANT HMM

### 2.1   MOTIVATIONS

The discriminant HMM/MLP theory as described above uses transition-based probabilities as the key building block for acoustic recognition. However, it is well known that estimating transitions accurately is a difficult problem (Glass 1988). Due to the inertia of the articulators, the boundaries between phones are blurred and overlapped in continuous speech. In our previous hybrid HMM/MLP system, targets were typically obtained by using a standard forced Viterbi alignment (segmentation). For a transition-based system as defined above, this procedure would thus yield rigid transition targets, which is not realistic.

Another problem related to the Viterbi-based training of the MLP presented in Figure 2 and used in Discriminant HMMs, is the lack of coverage of the input space during training. Indeed, during training (based on hard transitions), the MLP only processes inputs consisting of "correct" pairs of acoustic vectors and correct previous state, while in recognition the net should generalize to all possible combinations of

acoustic vectors and previous states, since all possible models and transitions will be hypothesized for each acoustic input. For example, some hypothesized inputs may correspond to an impossible condition that has thus never been observed, such as the acoustics of the temporal center of a vowel in combination with a previous state that corresponds to a plosive. It is unfortunately possible that the interpolative capabilities of the network may not be sufficient to give these "impossible" pairs a sufficiently low probability during recognition.

One possible solution to these problems is to use a full MAP algorithm to find transition probabilities at each frame for all possible transitions by a forward-backward-like algorithm (Liporace 1982), taking all possible paths into account.

## 2.2  PROBLEM FORMULATION

As described above, global maximum a posteriori training of HMMs should find the optimal parameter set $\Theta$ maximizing

$$\prod_{j=1}^{J} P(M_j|X_j, \Theta) \tag{1}$$

in which $M_j$ represents the Markov model associated with each training utterance $X_j$, with $j = 1, \ldots, J$.

Although in principle we could use a generalized back-propagation-like gradient procedure in $\Theta$ to maximize (1) (Bengio *et al.* 1992), an EM-like algorithm should have better convergence properties, and could preserve the statistical interpretation of the ANN outputs. In this case, training of the discriminant HMM by a global MAP criterion requires a solution to the following problem: given a trained MLP at iteration $t$ providing a parameter set $\Theta^t$ and, consequently, estimates of $P(q_n^\ell|x_n, q_{n-1}^k, \Theta^t)$, how can we determine new MLP targets that:

1. will be smooth estimates of conditional transition probabilities $q_{n-1}^k \to q_n^\ell$, $\forall k, \ell \in [1, K]$ and $\forall n \in [1, N]$,

2. when training the MLP for iteration $t+1$, will lead to new estimates of $\Theta^{t+1}$ and $P(q_n^\ell|x_n, q_{n-1}^k, \Theta^{t+1})$ that are guaranteed to incrementally increase the global posterior probability $P(M_i|X, \Theta)$?

In (Bourlard *et al.* 1994), we prove that a re-estimate of MLP targets that guarantee convergence to a local maximum of (1) is given by[1]:

$$P^*(q_n^\ell|x_n, q_{n-1}^k, M) = P(q_n^\ell|X, q_{n-1}^k, \Theta^t, M) \tag{2}$$

where we have estimated the left-hand side using a mapping from the previous state and the local acoustic data to the current state, thus making the estimator realizable by an MLP with a local acoustic window.[2] Thus, we will want to estimate

the transition probability conditioned on the *local* data (as MLP targets) by using
the transition probability conditioned on *all* of the data.

In (Bourlard *et al.* 1994), we further prove that alternating MLP target estimation
(the "estimation" step) and MLP training (the "maximization" step) is guaranteed
to incrementally increase (1) over $t$.[3] The remaining problem is to find an efficient
algorithm to express $P(q_n^\ell | X, q_{n-1}^k, M)$ in terms of $P(q_n^\ell | x_n, q_{n-1}^k)$ so that the next
iteration targets can be found. We have developed several approaches to this esti-
mation, some of which are described in (Bourlard *et al.* 1994). Currently, we are
implementing this with an efficient recursion that estimates the sum of all possible
paths in a model, for every possible transition at each possible time. From these
values we can compute the desired targets (2) for network training by

$$P(q_n^\ell | X, M, q_{n-1}^k) = \frac{P(M, q_n^\ell, q_{n-1}^k | X)}{\sum_j P(M, q_n^j, q_{n-1}^k | X)} \qquad (3)$$

## 2.3   REMAP TRAINING ALGORITHM

The general scheme of the REMAP training of hybrid HMM/MLP systems can be
summarized as follow:

1. Start from some initial net providing $P(q_n^\ell | x_n, q_{n-1}^k, \Theta^t)$, $t = 0$, $\forall$ possible
   $(k, \ell)$-pairs[4].

2. Compute MLP targets $P(q_n^\ell | X_j, q_{n-1}^k, \Theta^t, M_j)$ according to (3), $\forall$ training
   sentences $X_j$ associated with HMM $M_j$, $\forall$ possible $(k, \ell)$ state transition
   pairs in $M_j$ and $\forall x_n$, $n = 1, \ldots, N$ in $X_j$ (see next point).

3. For every $x_n$ in the training database, train the MLP to minimize the
   relative entropy between the outputs and targets. See (Bourlard *et al.*
   1994) for more details. This provides us with a new set of parameters $\Theta^t$,
   for $t = t + 1$.

4. Iterate from 2 until convergence.

This procedure is thus composed of two steps: an Estimation (E) step, correspond-
ing to step 2 above, and a Maximization (M) step, corresponding to step 3 above.
In this regards, it is reminiscent of the Estimation-Maximization (EM) algorithm
as discussed in (Dempster *et al.* 1977). However, in the standard EM algorithm,
the M step involves the actual maximization of the likelihood function. In a related
approach, usually referred to as Generalized EM (GEM) algorithm, the M step does
not actually maximize the likelihood but simply increases it (by using, e.g., a gra-
dient procedure). Similarly, REMAP increases the global posterior function during
the M step (in the direction of targets that actually maximize that global function),
rather than actually maximizing it. Recently, a similar approach was suggested for
mapping input sequences to output sequences (Bengio & Frasconi 1995).

but for one estimation-maximization iteration for which a complete MLP training will be
required.

[4]This can be done, for instance, by training up such a net from a hand-labeled database
like TIMIT or from some initial forward-backward estimator of equivalent local probabil-
ities (usually referred to as "gamma" probabilities in the Baum-Welch procedure).

| System | Error Rate |
|---|---|
| DHMM, pre-REMAP | 14.9% |
| 1 REMAP iteration | 13.6% |
| 2 REMAP iterations | 13.2% |

Table 1: Training and testing on continuous numbers, no syntax, no durational models.

## 3 EXPERIMENTS AND RESULTS

For testing our theory we chose the Numbers'93 corpus. It is a continuous speech database collected by CSLU at the Oregon Graduate Institute. It consists of numbers spoken naturally over telephone lines on the public-switched network (Cole *et al.* 1994). The Numbers'93 database consists of 2167 speech files of spoken numbers produced by 1132 callers. We used 877 of these utterances for training and 657 for cross-validation and testing (200 for cross-validation) saving the remaining utterances for final testing purposes. There are 36 words in the vocabulary, namely *zero, oh, 1, 2, 3,...,20, 30, 40, 50,...,100, 1000, a, and, dash, hyphen,* and *double.* All our nets have 214 inputs: 153 inputs for the acoustic features, and 61 to represent the previous state (one unit for every possible previous state, one state per phoneme in our case). The acoustic features are combined from 9 frames with 17 features each (RASTA-PLP8 + delta features + delta log gain) computed with an analysis window of 25 ms computed every 12.5 ms (overlapping windows) and with a sampling rate of 8 Khz. The nets have 200 hidden units and 61 outputs.

Our results are summarized in Table 1. The row entitled "DHMM, pre-REMAP" corresponds to a Discriminant HMM using the same training approach, with hard targets determined by the first system, and additional inputs to represent the previous state The improvement in the recognition rate as a result of REMAP iterations is significant at $p < 0.05$. However all the experiments were done using acoustic information alone. Using our (baseline) hybrid system under equal conditions, i.e., no duration information and no language information, we get 31.6% word error; adding the duration information back we get 12.4% word error. We are currently experimenting with enforcing minimum duration constraints in our framework.

## 4 CONCLUSIONS

In summary:

- We have a method for MAP training and estimation of sequences.

- This can be used in a new form of hybrid HMM/MLP. Note that recurrent nets or TDNNs could also be used. As with standard HMM/MLP hybrids, the network is used to estimate local posterior probabilities (though in this case they are conditional transition probabilities, that is, state probabilities conditioned on the acoustic data and the previous state). However, in the case of REMAP these nets are trained with probabilistic targets that are themselves estimates of local posterior probabilities.

- Initial experiments demonstrate a significant reduction in error rate for this process.

## Acknowledgments

We would like to thank Kristine Ma and Su-Lin Wu for their help with the Numbers'93 database. We also thank OGI, in particular to Ron Cole, for providing the database. We gratefully acknowledge the support of the Office of Naval Research, URI No. N00014-92-J-1617 (via UCB), the European Commission via ESPRIT project 20077 (SPRACH), and ICSI and FPMs in general for supporting this work.

## Footnotes

*Also affiliated with with Faculté Polytechnique de Mons, Mons, Belgium

[1]In most of the following, we consider only one particular training sequence $X$ associated with one particular model $M$. It is, however, easy to see that all of our conclusions remain valid for the case of several training sequences $X_j$, $j = 1, \ldots, J$. A simple way to look at the problem is to consider all training sequences as a single training sequence obtained by concatenating all the $X_j$'s with boundary conditions at every possible beginning and ending point.

[2]Note that, as done in our previous hybrid HMM/MLP systems, all conditional on $x_n$ can be replaced by $X_{n-c}^{n+d} = \{x_{n-c}, \ldots, x_n, \ldots, x_{n+d}\}$ to take some acoustic context into account.

[3]Note here that one "iteration" does not stand for one iteration of the MLP training

# References

BENGIO, Y., & P. FRASCONI. 1995. An input output HMM architecture. In *Advances in Neural Information Processing Systems*, ed. by G. Tesauro, D. Touretzky, & T. Leen, volume 7. Cambridge: MIT press.

——, R. DE MORI, G. FLAMMIA, & R. KOMPE. 1992. Global optimization of a neural network-hidden Markov model hybrid. *IEEE trans. on Neural Networks* 3.252–258.

BOURLARD, H., Y. KONIG, & N. MORGAN. 1994. REMAP: Recursive estimation and maximization of a posteriori probabilities, application to transition-based connectionist speech recognition. Technical Report TR-94-064, International Computer Science Institute, Berkeley, CA.

——, & N. MORGAN. 1994. *Connectionist Speech Recognition - A Hybrid Approach*. Kluwer Academic Publishers.

——, & C. J. WELLEKENS. 1989. Links between Markov models and multilayer perceptrons. In *Advances in Neural Information Processing Systems 1*, ed. by D.J. Touretzky, 502–510, San Mateo. Morgan Kaufmann.

COLE, R.A., M. FANTY, & T. LANDER. 1994. Telephone speech corpus development at CSLU. In *Proceedings Int'l Conference on Spoken Language Processing*, Yokohama, Japan.

DEMPSTER, A. P., N. M. LAIRD, & D. B. RUBIN. 1977. Maximum likelihood from incomplete data via the *EM* algorithm. *Journal of the Royal Statistical Society, Series B* 34.1–38.

GLASS, J. R., 1988. *Finding Acoustic Regularities in Speech Applications to Phonetic Recognition*. M.I.T dissertation.

KATAGIRI, S., C.H. LEE, & JUANG B.H. 1991. New discriminative training algorithms based on the generalized probabilistic decent method. In *Proc. of the IEEE Workshop on Neural Netwroks for Signal Processing*, ed. by B.H. Juang, S.Y. Kung, & C.A. Kamm, 299–308.

KONIG, Y., & N. MORGAN. 1994. Modeling dynamics in connectionist speech recognition - the time index model. In *Proceedings Int'l Conference on Spoken Language Processing*, 1523–1526, Yokohama, Japan.

LIPORACE, L. A. 1982. Maximum likelihood estimation for multivariate observations of markov sources. *IEEE Trans. on Information Theory* IT-28.729–734.

MORGAN, N., H. BOURLARD, S. GREENBERG, & H. HERMANSKY. 1994. Stochastic perceptual auditory-event-based models for speech recognition. In *Proceedings Int'l Conference on Spoken Language Processing*, 1943–1946, Yokohama, Japan.